# NEURAL NETWORKS FOR TEMPLATE MATCHING: APPLICATION TO REAL-TIME CLASSIFICATION OF THE ACTION POTENTIALS OF REAL NEURONS

Yiu-fai Wong†, Jashojiban Banik† and James M. Bower‡
†Division of Engineering and Applied Science
‡Division of Biology
California Institute of Technology
Pasadena, CA 91125

## ABSTRACT

Much experimental study of real neural networks relies on the proper classification of extracellulary sampled neural signals (i.e. action potentials) recorded from the brains of experimental animals. In most neurophysiology laboratories this classification task is simplified by limiting investigations to single, electrically well-isolated neurons recorded one at a time. However, for those interested in sampling the activities of many single neurons simultaneously, waveform classification becomes a serious concern. In this paper we describe and constrast three approaches to this problem each designed not only to recognize isolated neural events, but also to separately classify temporally overlapping events in real time. First we present two formulations of waveform classification using a neural network template matching approach. These two formulations are then compared to a simple template matching implementation. Analysis with real neural signals reveals that simple template matching is a better solution to this problem than either neural network approach.

## INTRODUCTION

For many years, neurobiologists have been studying the nervous system by using single electrodes to serially sample the electrical activity of single neurons in the brain. However, as physiologists and theorists have become more aware of the complex, nonlinear dynamics of these networks, it has become apparent that serial sampling strategies may not provide all the information necessary to understand functional organization. In addition, it will likely be necessary to develop new techniques which sample the activities of multiple neurons simultaneously[1]. Over the last several years, we have developed two different methods to acquire multineuron data. Our initial design involved the placement of many tiny microelectrodes individually in a tightly packed pseudo-floating configuration within the brain[2]. More recently we have been developing a more sophisticated approach which utilizes recent advances in silicon technology to fabricate multi-ported silicon based electrodes (Fig. 1). Using these electrodes we expect to be able to readily record the activity patterns of larger number of neurons.

As research in multi-single neuron recording techniques continue, it has become very clear that whatever technique is used to acquire neural signals from many brain locations, the technical difficulties associated with sampling, data compressing, storing, analyzing and interpreting these signals largely dwarf the development of the sampling device itself. In this report we specifically consider the need to assure that neural action potentials (also known as "spikes") on each of many parallel recording channels are correctly classified, which is just one aspect of the problem of post-processing multi-single neuron data. With more traditional single electrode/single neuron recordings, this task usually in-

volves passing analog signals through a Schmidt trigger whose output indicates the occurence of an event to a computer, at the same time as it triggers an oscilloscope sweep of the analog data. The experimenter visually monitors the oscilloscope to verify the accuracy of the discrimination as a well-discriminated signal from a single neuron will overlap on successive oscilloscope traces (Fig. 1c). Obviously this approach is impractical when large numbers of channels are recorded at the same time. Instead, it is necessary to automate this classification procedure. In this paper we will describe and contrast three approaches we have developed to do this.

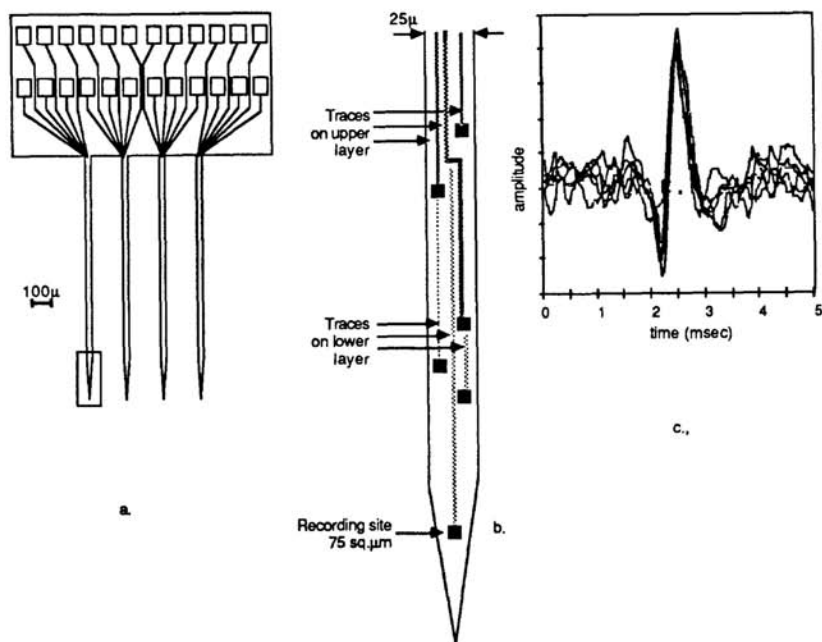

Fig. 1. Silicon probe being developed in our lababoratory for multi-single unit recording in cerebellar cortex. a) a complete probe; b) surface view of one recording tip; c) several superimposed neuronal action potentials recorded from such a silicon electrode in cerebellar cortex.

While our principal design objective is the assurance that neural waveforms are adequately discriminated on multiple channels, technically the overall objective of this research project is to sample from as many single neurons as possible. Therefore, it is a natural extention of our effort to develop a neural waveform classification scheme robust enough to allow us to distinguish activities arising from more than one neuron per recording site. To do this, however, we now not only have to determine that a particular signal is neural in origin, but also from which of several possible neurons it arose (see Fig. 2a). While in general signals from different neurons have different waveforms aiding in the classification, neurons recorded on the same channel firing simultaneously or nearly simultaneously will produce novel combination waveforms (Fig. 2b) which also need to be classified. It is this last complication which particularly

bedevils previous efforts to classify neural signals (For review see 5, also see 3-4). In summary, then, our objective was to design a circuit that would:

1. **distinguish different waveforms** even though neuronal discharges tend to be quite similar in shape (Fig. 2a);
2. **recognize the same waveform** even though unavoidable movements such as animal respiration often result in periodic changes in the amplitude of a recorded signal by moving the brain relative to the tip of the electrode;
3. be considerably **robust to recording noise** which variably corrupts all neural recordings (Fig. 2);
4. **resolve overlapping waveforms,** which are likely to be particularly interesting events from a neurobiological point of view;
5. provide **real-time performance** allowing the experimenter to detect problems with discrimination and monitor the progress of the experiment;
6. be **implementable in hardware** due to the need to classify neural signals on many channels simultaneously. Simply duplicating a software-based algorithm for each channel will not work, but rather, multiple, small, independent, and programmable hardware devices need to be constructed.

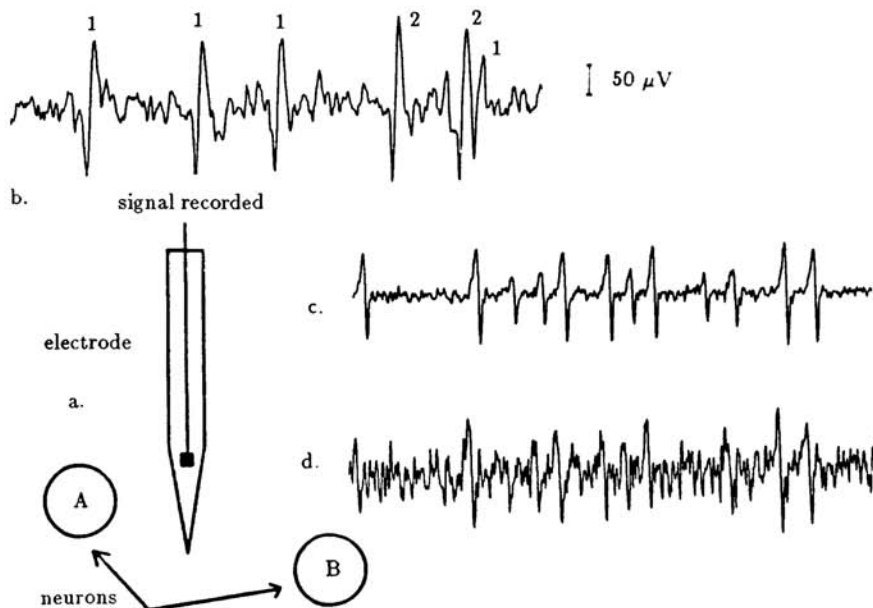

Fig. 2. a) Schematic diagram of an electrode recording from two neuronal cell bodies b) An actual multi-neuron recording. Note the similarities in the two waveforms and the overlapping event. c) and d) Synthesized data with different noise levels for testing classification algorithms (c: 0.3 NSR ; d: 1.1 NSR).

## METHODS

The problem of detecting and classifying multiple neural signals on single voltage records involves two steps. First, the waveforms that are present in a particular signal must be identified and the templates be generated; second, these waveforms must be detected and classified in ongoing data records. To accomplish the first step we have modified the principal component analysis procedure described by Abeles and Goldstein[3] to automatically extract templates of the distinct waveforms found in an initial sample of the digitized analog data. This will not be discussed further as it is the means of accomplishing the second step which concerns us here. Specifically, in this paper we compare three new approaches to ongoing waveform classification which deal explicitly with overlapping spikes and variably meet other design criteria outlined above. These approaches consist of a modified template matching scheme, and two applied neural network implementations. We will first consider the neural network approaches. On a point of nomenclature, to avoid confusion in what follows, the real neurons whose signals we want to classify will be referred to as "neurons" while computing elements in the applied neural networks will be called "Hopons."

**Neural Network Approach** — Overall, the problem of classifying neural waveforms can best be seen as an optimization problem in the presence of noise. Much recent work on neural-type network algorithms has demonstrated that these networks work quite well on problems of this sort[6-8]. In particular, in a recent paper Hopfield and Tank describe an A/D converter network and suggest how to map the problem of template matching into a similar context[8]. The energy functional for the network they propose has the form:

$$E = \frac{-1}{2} \sum_i \sum_j T_{ij} V_i V_j - \sum_i V_i I_i \tag{1}$$

where $T_{ij}$ = connectivity between Hopon $i$ and Hopon $j$, $V_i$ = voltage output of Hopon $i$, $I_i$ = input current to Hopon $i$ and each Hopon has a sigmoid input-output characteristic $V = g(u) = 1/(1 + exp(-au))$.

If the equation of motion is set to be:

$$du_i/dt = -\partial E/\partial V = \sum_j T_{ij} V_j + I_i \tag{1a}$$

then we see that $dE/dt = -\left(\sum_j T_{ij} V_j + I_i\right) dV/dt = -(du/dt)(dV/dt) = -g'(u)(du/dt)^2 \leq 0$. Hence $E$ will go to to a minimum which, in a network constructed as described below, will correspond to a proposed solution to a particular waveform classification problem.

**Template Matching using a Hopfield-type Neural Net** — We have taken the following approach to template matching using a neural network. For simplicity, we initially restricted the classification problem to one involving two waveforms and have accordingly constructed a neural network made up of two groups of Hopons, each concerned with discriminating one or the other waveform. The classification procedure works as follows: first, a Schmidt trigger

is used to detect the presence of a voltage on the signal channel above a set threshold. When this threshold is crossed, implying the presence of a possible neural signal, 2 msecs of data around the crossing are stored in a buffer (40 samples at 20 KHz). Note that biophysical limitations assure that a single real neuron cannot discharge more than once in this time period, so only one waveform of a particular type can occur in this data sample. Also, action potentials are of the order of 1 msec in duration, so the 2 msec window will include the full signal for single or overlapped waveforms. In the next step (explained later) the data values are correlated and passed into a Hopfield network designed to minimize the mean-square error between the actual data and the linear combination of different delays of the templates. Each Hopon in the set of Hopons concerned with one waveform represents a particular temporal delay in the occurrence of that waveform in the buffer. To express the network in terms of an energy function formulation: Let $x(t)$ = input waveform amplitude in the $t^{th}$ time bin, $s_j(t)$ = amplitude of the $j^{th}$ template, $V_{jk}$ denote if $s_j(t-k)(j^{th}$ template delayed by $k$ time bins)is present in the input waveform. Then the appropriate energy function is:

$$E = \frac{1}{2} \sum_t \left( x(t) - \sum_{j,k} V_{jk} s_j(t-k) \right)^2$$
$$- \frac{1}{2} \sum_{t,j,k} V_{jk}(V_{jk} - 1) s_j^2(t-k) \qquad (2)$$
$$+ \gamma \sum_{j,k_1 < k_2} V_{jk_1} V_{jk_2}$$

The first term is designed to minimize the mean-square error and specifies the best match. Since $V \in [0,1]$, the second term is minimized only when each $V_{jk}$ assumes values 0 or 1. It also sets the diagonal elements $T_{ij}$ to 0. The third term creates mutual inhibition among the processing nodes evaluating the same neuronal signal, which as described above can only occur once per sample.

Expanding and simplifying expression (2), the connection matrix is:

$$T_{(j_1,k_1),(j_2,k_2)} = \begin{cases} -\sum_t s_{j_1}(t-k_1) s_{j_2}(t-k_2) - \gamma \delta_{j_1 j_2} \\ 0 \end{cases} \qquad \text{if } j_1 = j_2, k_1 = k_2 \qquad (3a)$$

and the input current

$$I_{jk} = \sum_t x(t) s_j(t-k) - \frac{1}{2} \sum_t s_j^2(t-k) \qquad (3b)$$

As it can be seen, the inputs are the correlations between the actual data and the various delays of the templates subtracting a constant term.

**Modified Hopfield Network** — As documented in more detail in Fig. 3-4, the above full Hopfield-type network works well for temporally isolated spikes at moderate noise levels, but for overlapping spikes it has a local minima problem. This is more severe with more than two waveforms in the network.

Further, we need to build our network in hardware and the full Hopfield network is difficult to implement with current technology (see below). For these reasons, we developed a modified neural network approach which significantly reduces the necessary hardware complexity and also has improved performance. To understand how this works, let us look at the information contained in the quantities $T_{ij}$ and $I_{ij}$ (eq. 3a and 3b ) and make some use of them. These quantities have to be calculated at a pre-processing stage before being loaded into the Hopfield network. If after calculating these quantities, we can quickly rule out a large number of possible template combinations, then we can significantly reduce the size of the problem and thus use a much smaller (and hence more efficient) neural network to find the optimal solution. To make the derivation simple, we define slightly modified versions of $T_{ij}$ and $I_{ij}$ (eq. 4a and 4b) for two-template case.

$$T_{ij} = \sum_t s_1(t - i)s_2(t - j) \tag{4a}$$

$$I_{ij} = \sum_t x(t)\left[\frac{1}{2}s_1(t - i) + \frac{1}{2}s_2(t - j)\right] - \frac{1}{2}\sum_t s_1^2(t - i) - \frac{1}{2}\sum_t s_2^2(t - j) \tag{4b}$$

In the case of overlaping spikes the $T_{ij}$'s are the cross-correlations between $s_1(t)$ and $s_2(t)$ with different delays and $I_{ij}$'s are the cross-correlations between input $x(t)$ and weighted combination of $s_1(t)$ and $s_2(t)$. Now if $x(t) = s_1(t - i) + s_2(t - j)$ (i.e. the overlap of the first template with i time bin delay and the second template with j time bin delay), then $\Delta_{ij} = |T_{ij} - I_{ij}| = 0$. However in the presence of noise, $\Delta_{ij}$ will not be identically zero, but will equal to the noise, and if $\Delta_{ij} > \Delta T_{ij}$ (where $\Delta T_{ij} = |T_{ij} - T_{i'j'}|$ for $i \neq i'$ and $j \neq j'$) this simple algorithm may make unacceptable errors. A solution to this problem for overlapping spikes will be described below, but now let us consider the problem of classifying non-overlapping spikes. In this case, we can compare the input cross-correlation with the auto-correlations (eq. 4c and 4d).

$$T_i' = \sum_t s_1^2(t - i); \quad T_i'' = \sum_t s_2^2(t - i) \tag{4c}$$

$$I_i' = \sum_t x(t)s_1(t - i); \quad I_i'' = \sum_t x(t)s_2(t - i) \tag{4d}$$

So for non-overlapping cases, if $x(t) = s_1(t - i)$, then $\Delta_i' = |T_i' - I_i'| = 0$. If $x(t) = s_2(t - i)$, then $\Delta_i'' = |T_i'' - I_i''| = 0$.

In the absence of noise, then the minimum of $\Delta_{ij}, \Delta_i'$ and $\Delta_i''$ represents the correct classification. However, in the presence of noise, none of these quantities will be identically zero, but will equal the noise in the input $x(t)$ which will give rise to unacceptible errors. Our solution to this noise related problem is to choose a few minima (three have chosen in our case) instead of one. For each minimum there is either a known corresponding linear combination of templates for overlapping cases or a simple template for non-overlapping cases. A three neuron Hopfield-type network is then programmed so that each neuron corresponds to each of the cases. The input $x(t)$ is fed to this tiny network to resolve whatever confusion remains after the first step of "cross-correlation" comparisons. (Note: Simple template matching as described below can also be used in the place of the tiny Hopfield type network.)

**Simple Template Matching** — To evaluate the performances of these neural network approaches, we decided to implement a simple template matching scheme, which we will now describe. However, as documented below, this approach turned out to be the most accurate and require the least complex hardware of any of the three approaches. The first step is, again, to fill a buffer with data based on the detection of a possible neural signal. Then we calculate the difference between the recorded waveform and all possible combinations of the two previously identified templates. Formally, this consists of calculating the distances between the input $x(m)$ and all possible cases generated by all the combinations of the two templates.

$$d_{ij} = \sum_t |x(t) - \{s_1(t-i) + s_2(t-j)\}|$$

$$d_i' = \sum_t |x(t) - s_1(t-i)|; \quad d_i'' = \sum_t |x(t) - s_2(t-i)|$$

$$d_{min} = min(d_{ij}, d_i', d_i'')$$

$d_{min}$ gives the best fit of all possible combinations of templates to the actual voltage signal.

## TESTING PROCEDURES

To compare the performance of each of the three approaches, we devised a common set of test data using the following procedures. First, we used the principal component method of Abeles and Goldstein[3] to generate two templates from a digitized analog record of neural activity recorded in the cerebellum of the rat. The two actual spike waveform templates we decided to use had a peak-to-peak ratio of 1.375. From a second set of analog recordings made from a site in the cerebellum in which no action potential events were evident, we determined the spectral characteristics of the recording noise. These two components derived from real neural recordings were then digitally combined, the objective being to construct realistic records, while also knowing absolutely what the correct solution to the template matching problem was for each occurring spike. As shown in Fig. 2c and 2d, data sets corresponding to different noise to signal ratios were constructed. We also carried out simulations with the amplitudes of the templates themselves varied in the synthesized records to simulate waveform changes due to brain movements often seen in real recordings. In addition to two waveform test sets, we also constructed three waveform sets by generating a third template that was the average of the first two templates. To further quantify the comparisons of the three diffferent approaches described above we considered non-overlapping and overlapping spikes separately. To quantify the performance of the three different approaches, two standards for classification were devised. In the first and hardest case, to be judged a correct classification, the precise order and timing of two waveforms had to be reconstructed. In the second and looser scheme, classification was judged correct if the order of two waveforms was correct but timing was allowed to vary by $\pm 100$ $\mu$secs(i.e. $\pm 2$ time bins) which for most neurobiological applications is probably sufficient resolution. Figs. 3-4 compare the performance results for the three approaches to waveform classification implemented as digital simulations.

## PERFORMANCE COMPARISON

*Two templates – non-overlapping waveforms*: As shown in Fig. 3a, at low noise-to-signal ratios (NSRs below .2) each of the three approaches were comparable in performance reaching close to 100% accuracy for each criterion. As the ratio was increased, however the neural network implementations did less and less well with respect to the simple template matching algorithm with the full Hopfield type network doing considerably worse than the modified network. In the range of NSR most often found in real data (.2 - .4) simple template matching performed considerably better than either of the neural network approaches. Also it is to be noted that simple template matching gives an estimate of the goodness of fit betwwen the waveform and the closest template which could be used to identify events that should not be classified (e.g. signals due to noise).

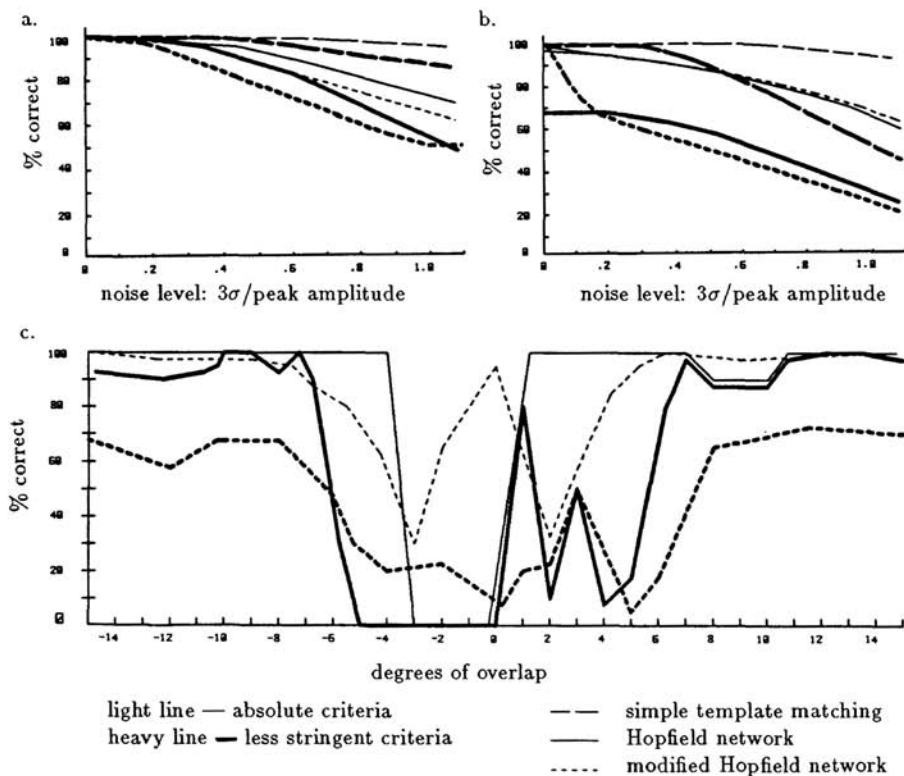

Fig. 3. Comparisons of the three approaches detecting two non-overlapping (a), and overlapping (b) waveforms, c) compares the performances of the neural network approaches for different degrees of waveform overlap.

*Two templates – overlapping waveforms*: Fig. 3b and 3c compare performances when waveforms overlapped. In Fig. 3b the serious local minima problem encountered in the full neural network is demonstrated as is the improved performance of the modified network. Again, overall performance in physi-

ological ranges of noise is clearly best for simple template matching. When the noise level is low, the modified approach is the better of the two neural networks due to the reliability of the correlation number which reflects the resemblence between the input data and the template. When the noise level is high, errors in the correlation numbers may exclude the right combination from the smaller network. In this case its performance is actually a little worse than the larger Hopfield network. Fig. 3c documents in detail which degrees of overlap produce the most trouble for the neural network approaches at average NSR levels found in real neural data. It can be seen that for the neural networks, the most serious problem is encountered when the delays between the two waveforms are small enough that the resulting waveform looks like the larger waveform with some perturbation.

*Three templates – overlapping and non-overlapping*: In Fig. 4 are shown the comparisons between the full Hopfield network approach and the simple template matching approach. For nonoverlapping waveforms, the performance of these two approaches is much more comparable than for the two waveform case (Fig. 4a), although simple template matching is still the optimal method. In the overlapping waveform condition, however, the neural network approach fails badly (Fig. 4b and 4c). For this particular application and implementation, the neural network approach does not scale well.

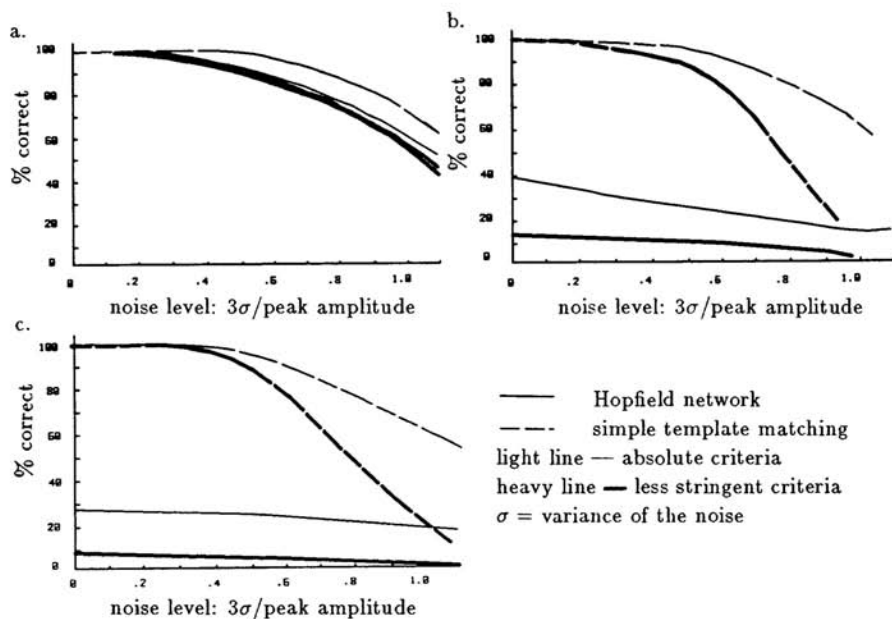

Fig. 4. Comparisons of performance for three waveforms. a) nonoverlapping waveforms; b) two waveforms overlapping; c) three waveforms overlapping.

## HARDWARE COMPARISONS

As described earlier, an important design requirement for this work was the ability to detect neural signals in analog records in real-time originating from

many simultaneously active sampling electrodes. Because it is not feasible to run the algorithms in a computer in real time for all the channels simultaneously, it is necessary to design and build dedicated hardware for each channel. To do this, we have decided to design VLSI implementations of our circuitry. In this regard, it is well recognized that large modifiable neural networks need very elaborate hardware implementations. Let us consider, for example, implementing hardwares for a two-template case for comparisons. Let $n =$ no. of neurons per template (one neuron for each delay of the template), $m =$ no. of iterations to reach the stable state (in simulating the discretized differential equation, with step size $= 0.05$), $l =$ no. of samples in a template $t_j(m)$. Then, the number of connections in the full Hopfield network will be $4n^2$. The total no. of synaptic calculations $= 4mn^2$. So, for two templates and $n = 16, m = 100, 4mn^2 = 102,400$. Thus building the full Hopfield-type network digitally requires a system too large to be put in a single VLSI chip which will work in real time. If we want to build an analog system, we need to have many $(O(4n^2))$ easily modifiable synapses. As yet this technology is not available for nets of this size. The modified Hopfield-type network on the other hand is less technically demanding. To do the preprocessing to obtain the minimum values we have to do about $n^2 = 256$ additions to find all possible $I_{ij}s$ and require 256 subtractions and comparisons to find three minima. The costs associated with doing input cross-correlations are the same as for the full neural network (i.e. $2nl = 768 (l = 24)$ multiplications). The saving with the modified approach is that the network used is small and fast (120 multiplications and 120 additions to construct the modifiable synapses, no. of synaptic calculations $= 90$ with $m = 10, n = 3$).

In contrast to the neural networks, simple template matching is simple indeed. For example, it must perform about $n^2 l + n^2 = 10,496$ additions and $n^2 = 256$ comparisons to find the minimum $d_{ij}$. Additions are considerably less costly in time and hardware than multiplications. In fact, because this method needs only addition operations, our preliminary design work suggests it can be built on a single chip and will be able to do the two-template classification in as little as 20 microseconds. This actually raises the possibility that with switching and buffering one chip might be able to service more than one channel in essentially real time.

## CONCLUSIONS

Template matching using a full Hopfield-type neural network is found to be robust to noise and changes in signal waveform for the two neural waveform classification problem. However, for a three-waveform case, the network does not perform well. Further, the network requires many modifiable connections and therefore results in an elaborate hardware implementation. The overall performance of the modified neural network approach is better than the full Hopfield network approach. The computation has been reduced largely and the hardware requirements are considerably less demanding demonstrating the value of designing a specific network to a specified problem. However, even the modified neural network performs less well than a simple template-matching algorithm which also has the simplest hardware implementation. Using the simple template matching algorithm, our simulations suggest it will be possible to build a two or three waveform classifier on a single VLSI chip using CMOS technology that works in real time with excellent error characteristics. Further, such a chip will be able to accurately classify variably overlapping

neural signals.

## REFERENCES

[1] G. L. Gerstein, M. J. Bloom, I. E. Espinosa, S. Evanczuk & M. R. Turner, IEEE Trans. Sys. Cyb. Man., SMC-13, 668(1983).
[2] J. M. Bower & R. Llinas, Soc. Neurosci. Abst., 9, 607(1983).
[3] M. Abeles & M. H. Goldstein, Proc. IEEE, 65, 762(1977).
[4] W. M. Roberts & D. K. Hartline, Brain Res., 94, 141(1976).
[5] E. M. Schmidt, J. of Neurosci. Methods, 12, 95(1984).
[6] J. J. Hopfield, Proc. Natl. Acad. Sci.(USA), 81, 3088(1984).
[7] J. J. Hopfield & D. W. Tank, Biol. Cybern., 52, 141(1985).
[8] D. W. Tank & J. J. Hopfield, IEEE Trans. Circuits Syst., CAS-33, 533(1986).

## ACKNOWLEDGEMENTS

We would like to acknowledge the contribution of Dr. Mark Nelson to the intellectual development of these projects and the able assistance of Herb Adams, Mike Walshe and John Powers in designing and constructing support equipment. This work was supported by NIH grant NS22205, the Whitaker Foundation and the Joseph Drown Foundation.